# Risk Aversion in Markov Decision Processes via Near-Optimal Chernoff Bounds

**Teodor Mihai Moldovan**
Department of Computer Science
University of California at Berkeley
Berkeley CA 94720, USA
moldovan@cs.berkeley.edu

**Pieter Abbeel**
Department of Computer Science
University of California at Berkeley
Berkeley CA 94720, USA
pabbeel@cs.berkeley.edu

## Abstract

The expected return is a widely used objective in decision making under uncertainty. Many algorithms, such as value iteration, have been proposed to optimize it. In risk-aware settings, however, the expected return is often not an appropriate objective to optimize. We propose a new optimization objective for risk-aware planning and show that it has desirable theoretical properties. We also draw connections to previously proposed objectives for risk-aware planing: minmax, exponential utility, percentile and mean minus variance. Our method applies to an extended class of Markov decision processes: we allow costs to be stochastic as long as they are bounded. Additionally, we present an efficient algorithm for optimizing the proposed objective. Synthetic and real-world experiments illustrate the effectiveness of our method, at scale.

## 1   Introduction

The *expected* return is often the objective function of choice in planning problems where outcomes not only depend on the actor's decisions but also on random events. Often expectations are the natural choice, as the law of large numbers guarantees that the average return over many independent runs will converge to the expectation. Moreover, the linearity of expectations can often be leveraged to obtain efficient algorithms.

Some games, however, can only be played once, either because they take a very long time (investing for retirement), because we are not given a chance to try again if we lose (skydiving, crossing the road), or because i.i.d. versions of the game are not available (stock market). In this setting, we can no longer take advantage of the law of large numbers to ensure that the return is close to its expectation with high probability, so the expected return might not be the best objective to optimize. If we were pessimistic, we might assume that everything that can go wrong will go wrong and try to minimize the losses under this assumption. This is called minmax optimization and is sometimes useful, but, most often, the resulting policies are overly cautious. A more balanced and general approach would include minmax optimization and expectation optimization, corresponding respectively to absolute risk aversion and risk ignorance, but would also allow a spectrum of policies between these extremes.

As a motivating example, consider buying tickets to fly to a very important meeting. Shorter travel time is preferable, but even more importantly, it would be disastrous if you arrived late. Some flights arrive on time more often than others, and the delays might be amplified if you miss connecting flights. With these risks in mind, would you rather take a route with an expected travel time of 12:21 and no further guarantees, or would you prefer a route that takes less than 16:19 with 99% probability? Our method produces these options when traveling from Shreveport Regional Airport (SHV) to Rafael Hernández Airport (BQN). According to historical flight data, if you chose the former alter-

native you could end up travelling for 22 hours with 8% probability. Another example comes from software quality assurance. Amazon.com requires its sub-services to report and optimize performance at the $99.9^{th}$ percentile, rather than in expectation, to make sure that all of its customers have a good experience, not just the majority [1]. In the economics literature, this percentile criterion is known as *value at risk* and has become a widely used measure of risk after the market crash of 1987 [2]. At the same time, the classical method for managing risk in investment is Markovitz portfolio optimization where the objective is to optimize expectation minus weighted variance. These examples suggest that proper risk-aware planning should allow a trade-off between expectation and variance, and, at the same time, should provide some guarantees about the probability of failure.

Risk-aware planning for Markov decision processes (MDPs) is difficult for two main reasons. First, optimizing many of the intuitive risk-aware objectives seems to be intractable computationally. Both mean minus variance optimization and percentile optimization for MDPs have been shown to be NP-hard in general [3, 4]. Consequently, we can only optimize relaxations of these objectives in practice. Second, it seems to be difficult to find an optimization objective which correctly models our intuition of risk awareness. Even though expectation, variance and percentile levels relate to risk awareness, optimizing them directly can lead to counterintuitive policies as illustrated recently in [3], for the case of mean minus variance optimization, and in the appendix of this paper, for percentile optimization.

Planning under uncertainty in MDPs is an old topic that has been addressed by many authors. The minmax objective has been proposed in [5, 6], which propose a dynamic programming algorithm for optimizing it efficiently. Unfortunately, minmax policies tend to be overly cautious. A number of methods have been proposed for relaxations of mean minus variance optimization [3, 7]. Percentile optimization has been shown to be tractable when dealing with ambiguity in MDP parameters [8, 9], and it has also been discussed in the context of risk [10, 11]. Our approach is closest to the line of work on exponential utility optimization [12, 13]. This problem can be solved efficiently and the resulting policies conform to our intuition of risk awareness. However, previous methods give no guarantees about probability of failure or variance. For an overview of previously used objectives for risk-aware planning in MDPs, see [14, 15].

Our method arises from approaching the problem in the context of probability theory. We observe connections between exponential utility maximization, Chernoff bounds, and cumulant generating functions, which enables formulating a new optimization objective for risk-aware planning. This new objective is essentially a re-parametrization of exponential utility, and inherits both the efficient optimization algorithms and the concordance to intuition about risk awareness. We show that optimizing the proposed objective includes, as limiting cases, both minmax and expectation optimization and allows interpolation between them. Additionally, we provide guarantees at a certain percentile level, and show connections to mean minus variance optimization.

Two experiments, one synthetic and one based on real-world data, support our theoretical guarantees and showcase the proposed optimization algorithms. Our largest MDP has 124791 state-action pairs—significantly larger than experiments in most past work on risk-aware planning. Our experiments illustrate the ability of our approach to—out of the exponentially many policies available—produce a family of policies that agrees with the human intuition of varying risk.

## 2   Background and Notation

An MDP consists of a state space $\mathcal{S}$, an action space $\mathcal{A}$, state transition dynamics, and a cost function $G$. Assume that, at time $t$, the system is in state $s_t \in \mathcal{S}$. Once the player chooses an action $a_t \in \mathcal{A}$, the system transitions stochastically to state $s_{t+1} \in \mathcal{S}$, with probability $p(s_{t+1}|s_t, a_t)$, and the player incurs a stochastic cost of $G^t(s_t, a_t, s_{t+1})$. The process continues for a number of time steps, $h$, called the *horizon*. We eventually care about the total cost obtained. We represent the player's strategy as a time dependent *policy*, which is a measure on the space of state-actions. Finally, we set the starting state to some fixed $s_0 \in \mathcal{S}$. Then, the objective is to "optimize" the *random variable* $J^h$, defined by $J^h := \sum_{t=0}^{h-1} G^t(S_t, A_t, S_{t+1})$. Traditionally, "optimizing" $J$ means minimizing its expected value, that is solving $\min_\pi E_{s,\pi}[J]$. The classical solution to this problem is to run *value*

*iteration*, summarized below:

$$q^{t+1}(s,a) := \sum_{s'} p_{s'|s,a} \left( G^t_{s,a,s'} + j^t(s') \right), \qquad j^t(s) := \min_a q^t(s,a) = \min_\pi E_{s,\pi}[J^t]$$

We will refer to policies obtained by standard value iteration as *expectimin* policies. We use upper case letters for random variables. We assume that the state-action space is finite and that sums with zero terms, for example $J^0$, are equal to zero. The notation $E_{s,\pi}$ signifies taking the expectation starting from $S_0 = s$, and following policy $\pi$. We assume that costs are upper bounded, that is there exists $j_M$ such that $J \leq j_M$ almost surely for any start state and any policy, and that the expected costs are finite. Finally, in this paper we will not consider discounting explicitly. If necessary, discounting can be introduced in one of two ways: either by adding a transition from every state, for all actions, to an absorbing "end game" state, with probability $\gamma$, or by setting a time dependent cost as $G^t_{\text{new}} = \gamma^t G^t_{\text{old}}$. Note that these two ways of introducing discounting are equivalent when optimizing the expected cost, but they can differ in the risk-aware setting we are considering. We refer the reader to [16] and [17] for further background on MDPs.

## 3    The Chernoff Functional as Risk-Aware Objective

We propose optimizing the following functional of the cost, which we call the *Chernoff functional* since it often appears in proving Chernoff bounds:

$$C^\delta_{s,\pi}[J] = \inf_{\theta > 0} \left( \theta \log E_{s,\pi} \left[ e^{J/\theta} \right] - \theta \log(\delta) \right). \tag{1}$$

First, note the total cost appears in the expression of the Chernoff functional as an exponential utility $(E_{s,\pi}[e^{J/\theta}])$. This shows that there is a strong connection between our method and exponential utility optimization. Specifically, all policies proposed by our algorithm, including the final solution, are optimal policies with respect to the exponential utility for some parameter. These policies are known to show risk-awareness in practice [12, 13], and our method inherits this property. In some sense, our proposed objective is a *re-parametrization* of exponential utility, which was obtained through its connections to Chernoff bounds and cumulant generating functions. The theorem below, which is one of the main contributions of this paper, provides more reasons for optimizing the Chernoff functional in the risk-aware setting. We will state and discuss the theorem here, but leave the proof for the appendix.

**Theorem 1.** *Let $\delta \in [0,1]$, and let $J$ be a random variable that has a cumulant generating function, that is $E \exp(J/\theta) < \infty$ for all $\theta > 0$. Then, the Chernoff functional of this random variable, $C^\delta[J]$, is well defined, and has the following properties:*

*(i)* $P(J \geq C^\delta[J]) \leq \delta$

*(ii)* $C^1[J] = \lim_{\theta \to \infty} \theta \log E[e^{J/\theta}] = E[J]$

*(iii)* $C^0[J] := \lim_{\delta \to 0} C^\delta[J] = \lim_{\theta \to 0} \theta \log E[e^{J/\theta}] = \sup\{j : P\{J \geq j\} > 0\} < \infty.$

*(iv)* $C^\delta[J] = E[J] + \sqrt{2 \log(1/\delta) \text{Var}[J]}$    *if $J$ is Gaussian.*

*(v) As $\delta \to 1$,*    $C^\delta[J] \approx E[J] + \sqrt{2 \log(1/\delta) \text{Var}[J]}$

*(vi)* $C^\delta[J]$ *is a smooth, decreasing function of $\delta$.*

*Proof sketch.* Property (i) is simply a Chernoff bound and follows by applying Markov's inequality to the random variable $e^{J/\theta}$. Property (iv) follows from the fact that all but the first two cumulants of Gaussian random variables are zero [18]. Properties (ii), (iii), (v) and (vi) follow from the following properties of *cumulant generating function*, $\log E e^{zJ}$, [18]:

(a) $\log E e^{zJ} = \sum_{i=1}^{\infty} z^i k_i / i!$    where $k_i$ are the *cumulants* [18], e.g. $k_1 = E[J], k_2 = \text{Var}[J]$.

(b) $\log E e^{zJ}$ as a function of $z \in \mathbb{R}$ is strictly convex, analytic and infinitely differentiable in a neighborhood of zero, if it is finite in that neighborhood.

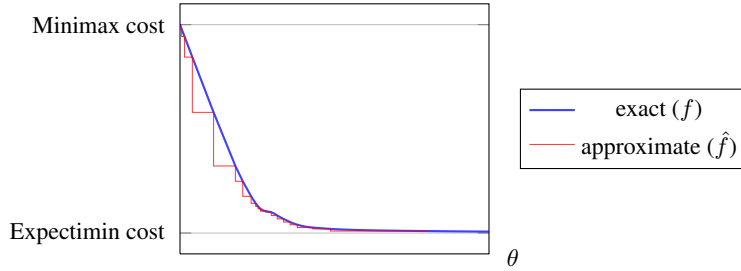

Figure 1: Plot showing the exact function $f$ defined in Equation 2 and the approximation that our algorithm constructs $\hat{f}$ for the Grid World MDP described in Section 5.1.

Properties (ii) and (iii) show that we can use the $\delta$ parameter to interpolate between the nominal policy, which ignores risk, at $\delta = 1$, and the minmax policy, which corresponds to extreme risk aversion, at $\delta = 0$. Property (i) shows that the value of the Chernoff functional is with probability at least $1 - \delta$ an upper bound on the cost obtained by following the corresponding Chernoff policy. These two observations suggests that by tuning $\delta$ from 0 to 1 we can find a family of risk-aware policies, in order of risk aversion. Our experiments support this hypothesis (Section 5).

Property (i) shows a connection between our approach and percentile optimization. Although we are not optimizing the $\delta$-percentile directly, our method provides guarantees about it. Properties (iv) and (v) show a connection between optimizing the Chernoff functional and mean minus variance optimization, which has been proposed before for risk-aware planning, but was found to be intractable in general [3]. Via property (v), we can optimize mean minus variance with a low weight on variance if we set $\delta$ close to 1. In the limit, this allows us to optimize the expectation, while breaking ties in favor of lower variance. Property (iv) show that we can optimize mean minus scaled standard deviation exactly if the total cost is Gaussian. Typically, this will not be the case, but, if the MDP is ergodic and the time horizon is large enough, the total cost will be close to Gaussian, by the central limit theorem. To see why this is true, note that, by the Markov property, costs between successive returns to the same state are i.i.d. random variables [19]. Our formulation ties into mean minus standard deviation optimization, which is of consistent dimensionality, unlike the classical mean minus variance objective.

## 4 Optimizing the Proposed Objective

Finding the policy that optimizes our proposed objective at a given risk level $\delta$ amounts to a joint optimization problem (Bellman optimality does not hold for our objective; see Appendix for discussion):

$$\min_{\pi} C^{\delta}_{s,\pi}[J] = \inf_{\theta > 0} \left( \theta \log \left( \min_{\pi} E_{s,\pi} \left[ e^{J/\theta} \right] \right) - \theta \log(\delta) \right) \tag{2}$$
$$= \inf_{\theta > 0} \left( f(\theta) - \theta \log(\delta) \right) \quad \text{where} \quad f(\theta) := \theta \log \left( \min_{\pi} E_{s,\pi} \left[ e^{J/\theta} \right] \right).$$

The inner optimization problem, the optimization over policies $\pi$, is simply exponential utility optimization, a classical problem that can be solved efficiently. For brevity, we will not discuss solutions to this problem and, instead, refer the readers to [12, 13]. The main difficulty is solving the outer optimization problem, over the scale variable $\theta$. Unfortunately, this problem is not convex and may have a large number of local minima. Our main algorithmic contribution consists of an approach for solving the outer (non-convex) optimization problem efficiently to some specified precision $\varepsilon$.

Based on Theorems 1 and 2 (below), we propose a method for finding the policy that minimizes the Chernoff functional, to precision $\varepsilon$, with worst case time complexity $O(h|S|^2|A|/\varepsilon)$. It is summarized in Algorithm 1. Our approach is to solve the optimization problem in (2) with an approximation of the function $f$ (Figure 1 shows a example plot of this function). The algorithm maintains such an approximation and improves it as needed up to a precision of $\varepsilon$. In practice we might want to run the algorithm for more than one setting of $\delta$ to find policies for the same planning task at different levels of risk aversion, say at $n$ different levels. Naively, the time complexity of doing this

---

**Algorithm 1** Near optimal Chernoff bound algorithm

---

$\hat{f} \leftarrow$ empty hash map                $\triangleright$ will store incremental approximation of $f$ defined in Eq. 2
$\hat{f}[0] \leftarrow f(0)$                                     $\triangleright$ minimax cost of the MDP
$\hat{f}[\infty] \leftarrow f(\infty)$                                 $\triangleright$ expectimin cost of the MDP
**for** $\theta \in \{1, 10, 100, \cdots\}$, **until** $\hat{f}[\infty] - \hat{f}[\theta] < \varepsilon$, **do**          $\triangleright$ find upper bound
    $\hat{f}[\theta] \leftarrow f(\theta)$                             $\triangleright$ exponential utility optimization
**for** $\theta \in \{1, 0.1, 0.01, \cdots\}$, **until** $\hat{f}[\theta] - \hat{f}[0] < \varepsilon$, **do**          $\triangleright$ find lower bound
    $\hat{f}[\theta] \leftarrow f(\theta)$                             $\triangleright$ exponential utility optimization
**repeat**
    $\theta^* \leftarrow \mathrm{argmin}\{\theta \in \mathrm{keys}(\hat{f}) : \hat{f}[\theta] - \theta \log(\delta)\}$,    $\triangleright$ $argmin$ over previously computed costs
    $\theta \leftarrow \left( \theta^* \cdot \min\{\theta > \theta^*, \theta \in \mathrm{keys}(\hat{f})\} \right)^{1/2}$       $\triangleright$ split interval at geometric mean
    $\hat{f}[\theta] \leftarrow f(\theta)$                           $\triangleright$ exponential utility optimization
**until** $\hat{f}[\theta^*] - \hat{f}[\theta] < \varepsilon$             $\triangleright$ until $\hat{f}$ is an $\varepsilon$-accurate approximation of $f$
**return** optimal exponential utility policy(MDP, $1/\theta^*$).

---

would be $O(nh|S|^2|A|/\varepsilon)$ but, fortunately, our function approximation can be reused between subsequent runs of the algorithm, saving computation time, so the total complexity will, in fact, be only $O(h|S|^2|A|/\varepsilon + n)$.

Properties (ii) and (iii) of Theorem 1 imply that $f(0)$ can be computed by minimax optimization and $f(\infty)$ can be computed by value iteration (expectimin optimization), which both have the same time complexity as exponential utility optimization: $O(h|S|^2|A|)$. Once we have computed these limits, the next step in the algorithm is finding some appropriate bounding interval, $[\theta_1, \theta_2]$, such that $f(0) - f(\theta_1) < \varepsilon$ and $f(\theta_2) - f(\infty) < \varepsilon$. We do this by first searching over $\theta = 1, .1, 10^{-2}, \cdots$, and, then, over $\theta = 1, 10, 10^2, \cdots$. For a given machine architecture, the number of $\theta$ values is bounded by the number format used in the implementation. For example, working with double precision floating-point numbers limits the number of $\theta$ evaluations to $2 \cdot 1023$, implied by the fact that exponents are only assigned 11 bits. In our experiments, this step takes 10-15 function evaluations. Now, for any given risk level, $\delta$, we will find $\theta^*$ that minimizes the objective, $f(\theta) - \theta \log(\delta)$, among those $\theta$ where we have already evaluated $f$. We will, then, evaluate $f$ at a new point: the geometric mean of $\theta^*$ and its closest neighbor to the right. We stop iterating when the function value at the new point is less than $\varepsilon$ away from the function value at $\theta^*$, and return the corresponding optimal exponential utility policy. Consequently, our algorithm evaluates $f$ at a subset of the points $\{\theta_1(\theta_2/\theta_1)^{i/n} : i = 0, \cdots, n\}$ where $n$ is a power of 2. Theorem 2 guarantees that to get an $\varepsilon$ guarantee for the accuracy of the optimization it suffices to perform $n(\varepsilon) = O(1/\varepsilon)$ evaluations of $f$, where we are now treating $\log(\delta_2) - \log(\delta_1)$ as a constant. Therefore, the number of functions evaluations is $O(1/\varepsilon)$, and, since the time complexity of every evaluation is $O(h|S|^2|A|)$, the total time complexity of the algorithm is $O(h|S|^2|A|/\varepsilon)$.

**Theorem 2.** *Consider the interval $0 < \theta_1 < \theta_2$ split up into $n$ sub-intervals by $\theta_i^n = \theta_1(\theta_2/\theta_1)^{i/n}$, and let $\hat{f}_n(\theta) := f(\max_{i \in 0 \cdots n}\{\theta_i^n < \theta\})$ be our piecewise constant approximation to the function $f(\theta)$ defined in Equation (2). Then, for a given approximation error $\varepsilon$ there exists $n(\varepsilon) = O((\log(\delta_2) - \log(\delta_1))/\varepsilon)$ such that $|\hat{f}_{n(\varepsilon)}(\theta) - f(\theta)| \leq \varepsilon$ for all $\theta \in [\theta_1, \theta_2]$.*

*Proof sketch.* The key insight when proving this theorem is bounding rate of change of $f$. We can immediately see that $f_\pi(\theta) := \theta \log E_{s,\pi}\left[e^{J/\theta}\right]$ is a convex function since it is the perspective transformation of a convex function, namely, the cumulant generating function of the total cost $J$. Additionally, Theorem 1 shows that $f_\pi$ is lower bounded by $E_{s,\pi}[J]$, assumed to be finite, which implies that $f_\pi$ is non-increasing. On the other hand, by directly differentiating the definition of $f_\pi$, we get that $\theta f_\pi'(\theta) = f_\pi(\theta) - E_{s,\pi}[Je^{J/\theta}]/E_{s,\pi}[e^{J/\theta}]$.

Since we assumed that the costs, $J$, are upper bounded, there exist a maximum cost $j_M$ such that $J \leq j_M$ almost surely for any starting state $s$, and any policy $\pi$. We have also shown that $f_\pi(\theta) \geq E_{s,\pi}[J] \geq j_m := \min_{\pi'} E_{s,\pi'}[J]$, so we conclude that $-(j_M - j_m)/\theta \leq f_\pi'(\theta) \leq 0$ for any policy, $\pi$. Now that we have bounded the derivative of $f_\pi$ we can see that the value of $f$ can not change too

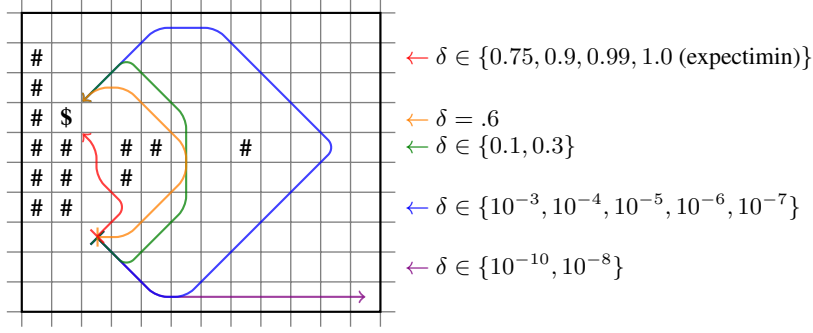

Figure 2: Chernoff policies for the Grid World MDP. See text for complete description. The colored arrows indicate the most likely paths under Chernoff policies for different values of $\delta$. The minimax policy ($\delta = 0$) acts randomly since it assumes that any action will lead to a trap.

much over an interval $[\theta_{i+1}^n, \theta_i^n]$. Let $\pi_i := \operatorname{argmin}_\pi f_\pi(\theta_i^n)$ and $\pi_{i+1} := \operatorname{argmin}_\pi f_\pi(\theta_{i+1}^n)$. Then:

$$0 \leq f(\theta_i^n) - f(\theta_{i+1}^n) = f_{\pi_i}(\theta_i^n) - f_{\pi_{i+1}}(\theta_{i+1}^n) \leq f_{\pi_{i+1}}(\theta_i^n) - f_{\pi_{i+1}}(\theta_{i+1}^n) \leq$$
$$\leq \max_{\theta_i^n \leq \theta \leq \theta_{i+1}^n} |f'_{\pi_{i+1}}(\theta)| \cdot (\theta_{i+1}^n - \theta_i^n) = -f'_{\pi_{i+1}}(\theta_i^n) \cdot (\theta_{i+1}^n - \theta_i^n) \leq$$
$$\leq (j_M - j_m) \cdot \frac{\theta_{i+1}^n - \theta_i^n}{\theta_i^n} = (j_M - j_m)\left( \left( \frac{\theta_2}{\theta_1} \right)^{1/n} - 1 \right), \tag{3}$$

where we first used the fact that $f_{\pi_i}(\theta_i^n) = \min_\pi f_\pi(\theta_i^n) \leq f_{\pi_{i+1}}(\theta_i^n)$, then the convexity of $f_{\pi_{i+1}}$ which implies that $f'_{\pi_{i+1}}$ is increasing, and, finally, our previous derivative bound. Our final goal is to find a value of $n(\varepsilon)$ such that the last expression in Equation 3 is less than $\varepsilon$. One can easily verify that the following $n(\varepsilon)$ satisfies this requirement (the detailed derivation appears in the Appendix):

$$n(\varepsilon) = \lceil (j_M - j_m)/\varepsilon \log{(\theta_2/\theta_1)} + \log{(\theta_2/\theta_1)} \rceil.$$

## 5 Experiments

We ran a number of experiments to test that our proposed objective indeed captures the intuitive meaning of risk-aware planning. The first experiment models a situation where it is immediately obvious what the family of risk-aware policies should be. We show that optimizing the Chernoff functional with increasing values of $\delta$ produces the intuitively correct family of policies. The second experiment shows that our method can be applied successfully to a large scale, real world problem, where it is difficult to immediately "see" the risk-aware family of policies.

Our experiments empirically confirm some of the properties of the Chernoff functional proven in Theorem 1: the probability that the return is lower than the value of the Chernoff policy at level $\delta$ is always less than $\delta$, setting $\delta = 1$ corresponds to optimizing the expected return with the added benefit of breaking ties in favor of lower variance, and setting $\delta = 0$ leads to the minmax policy whenever it is defined. Additionally, we observed that policies at lower risk levels, $\delta$, tend to have lower expectation but also lower variance, if the structure of the problem allows it. Generally, the probability of extremely bad outcomes decreases as we lower $\delta$.

### 5.1 Grid world

We first tested our algorithm on the Grid-World MDP (Figure 2). It models an obstacle avoidance problem with stochastic dynamics. Each state corresponds to a square in the grid, and the actions, {N, NE, E, SE, S, SW, W, NW}, typically cause a move in the respective direction. In unmarked squares, the actor's intention is executed with probability .93. Each of the seven remaining actions might be executed instead, each with probability 0.01. Squares marked with $ and # are absorbing states. The former gives a reward of 35 when entered, and the latter gives a penalty of 35. Any other state transitions cost 1. The horizon is 35. To make the problem finite, we simply set the

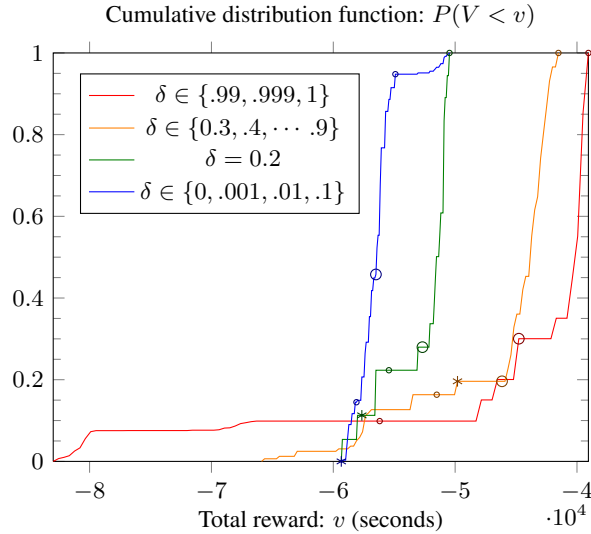

δ ∈ {.99, .999, 1.0 (expectimin)}:

15:45 SHV - DFW 16:45
18:25 DFW - MCO 21:50
23:15 MCO - BQN 02:46

δ ∈ {.3, .4, .5, .6, .7, .8, .9}:

10:46 SHV - ATL 13:31
14:10 ATL - EWR 16:30
18:00 EWR - BQN 23:00

δ = 0.2:

12:35 SHV - DFW 13:30
18:25 DFW - MCO 21:50
23:15 MCO - BQN 02:46

δ ∈ {0 (minimax) , .001, .01, .1}:

12:35 SHV - DFW 13:30
14:25 DFW - MSY 15:50
17:50 MSY - JFK 21:46
23:40 JFK - BQN 04:20

Cumulative distribution function: $P(V < v)$

(a) Paths under Chernoff policies assuming all flight arrive on time, shown using International Air Transport Association (IATA) airport codes.

(b) Cumulative distribution functions of rewards (equals minus cost) under Chernoff policies at different risk levels. The asterisk (*) indicates the value of the policy. The big O indicates the expected reward and the small o's correspond to expectation plus-minus standard deviation. 10000 samples.

Figure 3: Chernoff policies to travel from Shreveport Regional Airport (SHV) to Rafael Hernández Airport (BQN) at different risk levels.

probability of all transitions outside the grid boundary to zero, and re-normalize. We set the precision to $\varepsilon = 1$. With this setting, our algorithm performed exponential utility optimization for 97 different parameters when planning for 14 values of the risk level $\delta$. For low values of $\delta$, the algorithm behaves cautiously, preferring longer, but safer routes. For higher values of $\delta$, the algorithm is willing to take shorter routes, but also accepts increasing amounts of risk.

## 5.2 Air travel planning

The aerial travel planning MDP (Figure 3) illustrates that our method applies to real-world problems at a large scale. It models the problem of buying airplane tickets to travel between two cities, when you care only about reaching the destination in a reliable amount of time. We assume that, if you miss a connecting flight due to delays, the airline will re-issue a ticket for the route of your choice leading to the original destination. In this case, a cautious traveler will consider a number of aspects: choosing flights that usually arrive on time, choosing longer connection times and making sure that, in case of a missed connection, there are good alternative routes.

In our implementation, the state space consists of pairs of all airports and times when flights depart from those airports. At every state there are two actions: either take the flight that departs at that time, or wait. The total number of state-action pairs is 124791. To keep the horizon low, we introduce enough wait transitions so that it takes no more than 10 transitions to wait a whole day in the busiest airport (about 1000 flights per day) and we set the horizon at 100. Costs are deterministic and correspond to the time difference between the scheduled departure time of the first flight and the arrival time. We compute transition probabilities based on historical data, available from the Office of Airline Information, Bureau of Transportation Statistics, at http://www.transtats.bts.gov/. Particularly, we have used on-time statistics for February 2011. Airlines often try to conceal statistics for flights with low on-time performance by slightly changing departure times and flight numbers. Sometimes, they do this every week. Consequently, we first clustered together all flights with the same origin and destination that were scheduled to depart within 15 minutes of each other, under the assumption they would have the same on-time statistics. We, then, remove all clusters with fewer than 7 recorded flights, since these usually correspond to incidental flights.

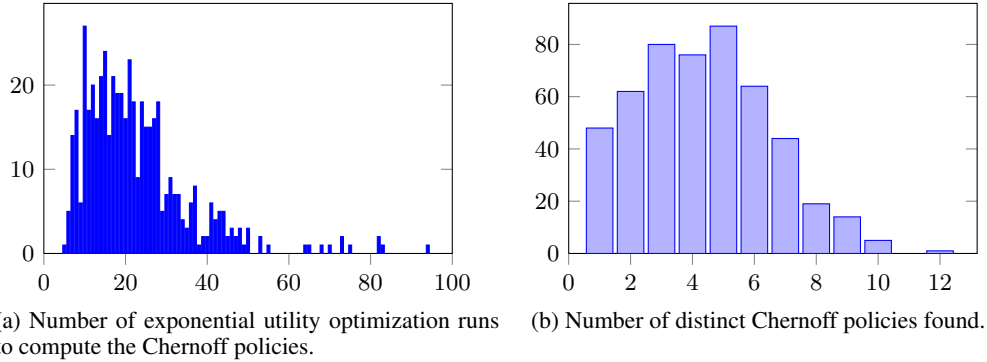

(a) Number of exponential utility optimization runs to compute the Chernoff policies.

(b) Number of distinct Chernoff policies found.

Figure 4: Histograms demonstrating the efficiency and relevance of our algorithm on $500$ randomly chosen origin - destination airport pairs, at $15$ risk levels.

To test our algorithm on this problem, we randomly chose $500$ origin - destination airport pairs and computed the Chernoff policies for risk levels: $\delta \in \{1.0, .999, .99, .9, .8, \cdots, .1, 0.01, 0.001, 0.0\}$, and precision $\varepsilon = 10$ minutes. Figure 3 shows the resulting policies and corresponding cost (travel time) histograms for one such randomly chosen route. To address the question of computational efficiency, Figure 4a shows a histogram of the total number of different parameters for which our algorithm ran exponential utility optimization. To address the question of relevance, Figure 4b shows the number of distinct Chernoff policies found among the risk levels. Two policies, $\pi$ and $\pi'$, are considered distinct if the total variation distance of the induced state - action occupation measures is more than $10^{-6}$; that is, if there exists $t$, $s$, and $a$ such that $|P_\pi\{S_t = s, A_t = a\} - P_{\pi'}\{S_t = s, A_t = a\}| \geq 10^{-6}$. For most origin - destination pairs we found a rich spectrum of distinct policies, but there are also cases where all the Chernoff policies are identical or only the expectimax and minimax policies differ.

Many air travel routes exhibit only two phases mainly because they connect small airports where only one or two flights of the type we consider land or take off per day. Consequently there will be few policies to choose from in these cases. In our experiment, we chose $200$ origin and destination pairs at random and, of these, $72$ routes show only two phases. In $41$ of these cases, either the origin or the destination airport serves only one or two flights per day total. Only $9$ of the two-phase routes connect airports which both serve more than $10$ flights per day total, and, of course, not all of these flight will help reach the destination. Thus, typically the reason we see only two phases is that the choice of policies is very limited. Additionally, airlines have an incentive to provide sufficient margin such that passengers can make connections and they don't have to re-ticket them. That is, they tend to set up routes such that, even in a worse than average scenario, the original route will tend to succeed.

## 6 Conclusion

We proposed a new optimization objective for risk-aware planning called the Chernoff functional. Our objective has a free parameter $\delta$ that can be used to interpolate between the nominal policy, which ignores risk, at $\delta = 1$, and the minmax policy, which corresponds to extreme risk aversion, at $\delta = 0$. The value of the Chernoff functional is with probability at least $1 - \delta$ an upper bound on the cost incurred by following the corresponding Chernoff policy. We established a close connection between optimizing the Chernoff functional and mean minus variance optimization, which has been proposed before for risk-aware planning, but was found to be intractable in general. We also establish a close connection with optimization of mean minus scaled standard deviation.

We proposed an efficient algorithm that optimizes the Chernoff functional to any desired accuracy $\varepsilon$ requiring $O(1/\varepsilon)$ runs of exponential utility optimization. Our experiments illustrate the capability of our approach to recover a spread of policies in the spectrum from risk neutral to minmax requiring a running time that was on average about ten times the running of value iteration.

## References

[1] G. DeCandia, D. Hastorun, M. Jampani, G. Kakulapati, A. Lakshman, A. Pilchin, S. Sivasub-ramanian, P. Vosshall, and W. Vogels. Dynamo: amazon's highly available key-value store. *ACM SIGOPS Operating Systems Review*, 41(6):205–220, 2007.

[2] Philippe Jorion. *Value at risk: the new benchmark for managing financial risk*, volume 1. McGraw-Hill Professional, 2007.

[3] Shie Mannor and John N. Tsitsiklis. Mean-Variance Optimization in Markov Decision Processes. In *Proceedings of the 28 International Conference on Machine Learning*, 2011.

[4] Erick Delage and Shie Mannor. Percentile optimization in uncertain Markov decision processes with application to efficient exploration. *ICML; Vol. 227*, page 225, 2007.

[5] Jay K. Satia and Roy E. Lave Jr. Markovian Decision Processes with Uncertain Transition Probabilities. *Operations Research*, 21(3):728–740, 1973.

[6] Matthias Heger. Consideration of risk in reinforcement learning. In *Proceedings of the 11th International Machine Learning Conference (1994)*, pages 105–111. Morgan Kaufmann, 1994.

[7] Steve Levitt and Adi Ben-Israel. On Modeling Risk in Markov Decision Processes. *Optimization and Related Topics*, pages 27–41, 2001.

[8] Erick Delage and Shie Mannor. Percentile Optimization for Markov Decision Processes with Parameter Uncertainty. *Operations Research*, 58(1):203–213, 2010.

[9] Arnab Nilim and Laurent El Ghaoui. Robust Control of Markov Decision Processes with Uncertain Transition Matrices. *Operations Research*, 53(5):780–798, 2005.

[10] M. Bouakiz and Y. Kebir. Target-level criterion in Markov decision processes. *Journal of Optimization Theory and Applications*, 86(1):1–15, July 1995.

[11] Congbin Wu and Yuanlie Lin. Minimizing Risk Models in Markov Decision Processes with Policies Depending on Target Values. *Journal of Mathematical Analysis and Applications*, 231(1):47–67, 1999.

[12] S.I. Marcus, E. Fernández-Gaucherand, D. Hernández-Hernandez, S. Coraluppi, and P. Fard. Risk sensitive Markov decision processes. *Systems and Control in the Twenty-First Century*, 29:263–281, 1997.

[13] VS Borkar and SP Meyn. Risk-sensitive optimal control for Markov decision processes with monotone cost. *Mathematics of Operations Research*, 27(1):192–209, 2002.

[14] B. Defourny, D. Ernst, and L. Wehenkel. Risk-aware decision making and dynamic programming. In *NIPS 2008 Workshop on Model Uncertainty and Risk in RL*, 2008.

[15] Yann Le Tallec. *Robust, Risk-Sensitive, and Data-driven Control of Markov Decision Processes*. PhD thesis, Massachusetts Institute of Technology, 2007.

[16] Richard S. Sutton and Andrew G. Barto. *Reinforcement learning: an introduction*. MIT Press, 1998.

[17] Dimitri P. Bertsekas and John N. Tsitsiklis. *Neuro-Dynamic Programming*. Athena Scientific, October 1996.

[18] J. F. Kenney and E. S. Keeping. Cumulants and the cumulant-generating function, additive property of cumulants, and Sheppard's correction. In *Mathematics of Statistics*, chapter 4.10-4.12, pages 77–82. Van Nostrand, Princeton, NJ, 2 edition, 1951.

[19] Richard Durrett. *Probability: Theory and Examples*. Cambridge University Press, 2010.

